# Reinforcement Learning for Continuous Stochastic Control Problems

**Rémi Munos**
CEMAGREF, LISC, Parc de Tourvoie,
BP 121, 92185 Antony Cedex, FRANCE.
Remi.Munos@cemagref.fr

**Paul Bourgine**
Ecole Polytechnique, CREA,
91128 Palaiseau Cedex, FRANCE.
Bourgine@poly.polytechnique.fr

## Abstract

This paper is concerned with the problem of Reinforcement Learning (RL) for continuous state space and time stochastic control problems. We state the Hamilton-Jacobi-Bellman equation satisfied by the value function and use a Finite-Difference method for designing a convergent approximation scheme. Then we propose a RL algorithm based on this scheme and prove its convergence to the optimal solution.

## 1 Introduction to RL in the continuous, stochastic case

The objective of RL is to find -thanks to a reinforcement signal- an optimal strategy for solving a dynamical control problem. Here we sudy the continuous time, continuous state-space stochastic case, which covers a wide variety of control problems including target, viability, optimization problems (see [FS93], [KP95]) for which a formalism is the following. The evolution of the *current state* $x(t) \in \bar{O}$ (the *state-space*, with $O$ open subset of $\mathbb{R}^d$), depends on the *control* $u(t) \in U$ (compact subset) by a stochastic differential equation, called the *state dynamics*:

$$dx = f(x(t), u(t))dt + \sigma(x(t), u(t))dw \qquad (1)$$

where $f$ is the local drift and $\sigma.dw$ (with $w$ a brownian motion of dimension $r$ and $\sigma$ a $d \times r$-matrix) the stochastic part (which appears for several reasons such as lake of precision, noisy influence, random fluctuations) of the diffusion process.

For initial state $x$ and control $u(t)$, (1) leads to an infinity of possible trajectories $x(t)$. For some trajectory $x(t)$ (see figure 1), let $\tau$ be its *exit time* from $\bar{O}$ (with the convention that if $x(t)$ always stays in $\bar{O}$, then $\tau = \infty$). Then, we define the *functional J* of initial state $x$ and control $u(.)$ as the expectation for all trajectories of the discounted cumulative reinforcement :

$$J(x; u(.)) = E_{x,u(.)} \left\{ \int_0^\tau \gamma^t r(x(t), u(t))dt + \gamma^\tau R(x(\tau)) \right\}$$

where $r(x, u)$ is the *running reinforcement* and $R(x)$ the *boundary reinforcement*. $\gamma$ is the *discount factor* $(0 \leq \gamma < 1)$. In the following, we assume that $f, \sigma$ are of class $C^2$, $r$ and $R$ are Lipschitzian (with constants $L_r$ and $L_R$) and the boundary $\partial O$ is $C^2$.

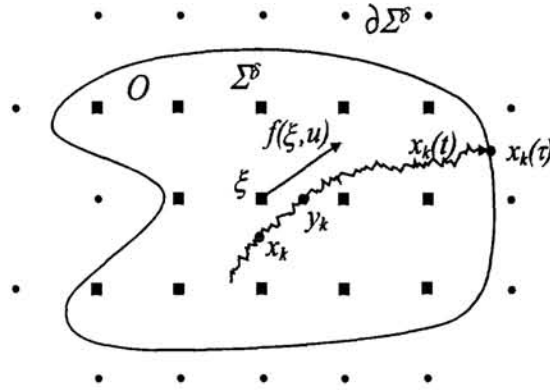

Figure 1: The state space, the discretized $\Sigma^\delta$ (the square dots) and its frontier $\partial \Sigma^\delta$ (the round ones). A trajectory $x_k(t)$ goes through the neighbourhood of state $\xi$.

RL uses the method of Dynamic Programming (DP) which generates an optimal (feed-back) control $u^*(x)$ by estimating the *value function* (VF), defined as the maximal value of the functional $J$ as a function of initial state $x$ :

$$V(x) = \sup_{u(.)} J(x; u(.)). \tag{2}$$

In the RL approach, the state dynamics is unknown from the system ; the only available information for learning the optimal control is the reinforcement obtained at the current state. Here we propose a model-based algorithm, i.e. that learns on-line a model of the dynamics and approximates the value function by successive iterations.

*Section 2* states the Hamilton-Jacobi-Bellman equation and use a Finite-Difference (FD) method derived from Kushner [Kus90] for generating a convergent approximation scheme. In *section 3*, we propose a RL algorithm based on this scheme and prove its convergence to the VF in *appendix A*.

## 2   A Finite Difference scheme

Here, we state a second-order nonlinear differential equation (obtained from the DP principle, see [FS93]) satisfied by the value function, called the *Hamilton-Jacobi-Bellman* equation.

Let the $d \times d$ matrix $a = \sigma.\sigma'$ (with $'$ the transpose of the matrix). We consider the *uniformly parabolic* case, i.e. we assume that there exists $c > 0$ such that $\forall x \in \overline{O}, \forall u \in U, \forall y \in \mathbb{R}^d, \sum_{i,j=1}^{d} a_{ij}(x,u) y_i y_j \geq c||y||^2$. Then $V$ is $C^2$ (see [Kry80]). Let $V_x$ be the gradient of $V$ and $V_{x_i x_j}$ its second-order partial derivatives.

**Theorem 1 (Hamilton-Jacobi-Bellman)** *The following HJB equation holds :*

$$V(x) \ln \gamma + \sup_{u \in U} \left[ r(x,u) + V_x(x).f(x,u) + \tfrac{1}{2} \sum_{i,j=1}^{n} a_{ij} V_{x_i x_j}(x) \right] = 0 \ \text{for } x \in O$$

*Besides, $V$ satisfies the following boundary condition : $V(x) = R(x)$ for $x \in \partial O$.*

**Remark 1** *The challenge of learning the VF is motivated by the fact that from $V$, we can deduce the following optimal feed-back control policy:*

$$u^*(x) \in \arg\sup_{u \in U} \left[ r(x,u) + V_x(x).f(x,u) + \tfrac{1}{2} \sum_{i,j=1}^{n} a_{ij} V_{x_i x_j}(x) \right]$$

In the following, we assume that $O$ is bounded. Let $e_1, ..., e_d$ be a basis for $\mathbb{R}^d$. Let the positive and negative parts of a function $\phi$ be : $\phi^+ = \max(\phi, 0)$ and $\phi^- = \max(-\phi, 0)$. For any *discretization step* $\delta$, let us consider the lattices : $\delta \mathbb{Z}^d = \left\{ \delta . \sum_{i=1}^{d} j_i e_i \right\}$ where $j_1, ..., j_d$ are any integers, and $\Sigma^\delta = \delta \mathbb{Z}^d \cap O$. Let $\partial \Sigma^\delta$, the *frontier* of $\Sigma^\delta$ denote the set of points $\{ \xi \in \delta \mathbb{Z}^d \setminus O$ such that at least one adjacent point $\xi \pm \delta e_i \in \Sigma^\delta \}$ (see figure 1).

Let $U^\delta \subset U$ be a finite control set that approximates $U$ in the sense: $\delta \leq \delta' \Rightarrow U^{\delta'} \subset U^\delta$ and $\overline{\cup_\delta U^\delta} = U$. Besides, we assume that: $\forall i = 1..d$,

$$a_{ii}(x,u) - \sum_{j \neq i} |a_{ij}(x,u)| \geq 0. \tag{3}$$

By replacing the gradient $V_x(\xi)$ by the forward and backward first-order finite-difference quotients: $\Delta_{x_i}^{\pm} V(\xi) = \tfrac{1}{\delta} [V(\xi \pm \delta e_i) - V(\xi)]$ and $V_{x_i x_j}(\xi)$ by the second-order finite-difference quotients:

$$\Delta_{x_i x_i} V(\xi) = \tfrac{1}{\delta^2} [V(\xi + \delta e_i) + V(\cdot - \delta e_i) - 2V(\xi)]$$

$$\Delta_{x_i x_j}^{\pm} V(\xi) = \tfrac{1}{2\delta^2} [V(\xi + \delta e_i \pm \delta e_j) + V(\xi - \delta e_i \mp \delta e_j) \\ - V(\xi + \delta e_i) - V(\xi - \delta e_i) - V(\xi + \delta e_j) - V(\xi - \delta e_j) + 2V(\xi)]$$

in the HJB equation, we obtain the following : for $\xi \in \Sigma^\delta$,

$$V^\delta(\xi) \ln \gamma + \sup_{u \in U^\delta} \left\{ r(\xi,u) + \sum_{i=1}^{d} \left[ f_i^+(\xi,u).\Delta_{x_i}^+ V^\delta(\xi) - f_i^-(\xi,u).\Delta_{x_i}^- V^\delta(\xi) \right. \right.$$
$$\left. \left. + \tfrac{a_{ii}(\xi,u)}{2} \Delta_{x_i x_i} V(\xi) + \sum_{j \neq i} \left( \tfrac{a_{ij}^+(\xi,u)}{2} \Delta_{x_i x_j}^+ V(\xi) - \tfrac{a_{ij}^-(\xi,u)}{2} \Delta_{x_i x_j}^- V(\xi) \right) \right] \right\} = 0$$

Knowing that $(\Delta t \ln \gamma)$ is an approximation of $(\gamma^{\Delta t} - 1)$ as $\Delta t$ tends to 0, we deduce :

$$V^\delta(\xi) = \sup_{u \in U^\delta} \left[ \gamma^{\tau(\xi,u)} \sum_{\zeta \in \Sigma^\delta} p(\xi,u,\zeta) V^\delta(\zeta) + \tau(\xi,u) r(\xi,u) \right] \tag{4}$$

$$\text{with } \tau(\xi,u) = \frac{\delta^2}{\sum_{i=1}^{d} \left[ \delta |f_i(\xi,u)| + a_{ii}(\xi,u) - \tfrac{1}{2} \sum_{j \neq i} |a_{ij}(\xi,u)| \right]} \tag{5}$$

which appears as a DP equation for some finite Markovian Decision Process (see [Ber87]) whose state space is $\Sigma^\delta$ and probabilities of transition :

$$p(\xi,u,\xi \pm \delta e_i) = \tfrac{\tau(\xi,u)}{2\delta^2} \left[ 2\delta |f_i^\pm(\xi,u)| + a_{ii}(\xi,u) - \sum_{j \neq i} |a_{ij}(\xi,u)| \right],$$

$$p(\xi,u,\xi + \delta e_i \pm \delta e_j) = \tfrac{\tau(\xi,u)}{2\delta^2} a_{ij}^\pm(\xi,u) \text{ for } i \neq j, \tag{6}$$

$$p(\xi,u,\xi - \delta e_i \pm \delta e_j) = \tfrac{\tau(\xi,u)}{2\delta^2} a_{ij}^\mp(\xi,u) \text{ for } i \neq j,$$

$$p(\xi,u,\zeta) = 0 \text{ otherwise.}$$

Thanks to a contraction property due to the discount factor $\gamma$, there exists a unique solution (the fixed-point) $V^\delta$ to equation (4) for $\xi \in \Sigma^\delta$ with the boundary condition $V^\delta(\xi) = R(\xi)$ for $\xi \in \partial \Sigma^\delta$. The following theorem (see [Kus90] or [FS93]) insures that $V^\delta$ is a convergent approximation scheme.

**Theorem 2 (Convergence of the FD scheme)** $V^\delta$ *converges to $V$ as $\delta \downarrow 0$ :*

$$\lim_{\substack{\delta \downarrow 0 \\ \xi \to x}} V^\delta(\xi) = V(x) \text{ uniformly on } \overline{O}$$

**Remark 2** *Condition (3) insures that the $p(\xi, u, \zeta)$ are positive. If this condition does not hold, several possibilities to overcome this are described in [Kus90].*

## 3    The reinforcement learning algorithm

Here we assume that $f$ is bounded from below. As the state dynamics ($f$ and $a$) is unknown from the system, we approximate it by building a *model* $\widetilde{f}$ and $\widetilde{a}$ from samples of trajectories $x_k(t)$ : we consider series of successive states $x_k = x_k(t_k)$ and $y_k = x_k(t_k + \tau_k)$ such that :

- $\forall t \in [t_k, t_k + \tau_k], \quad x(t) \in N(\xi)$ neighbourhood of $\xi$ whose diameter is inferior to $k_N.\delta$ for some positive constant $k_N$,

- the control $u$ is constant for $t \in [t_k, t_k + \tau_k]$,

- $\tau_k$ satisfies for some positive $k_1$ and $k_2$,

$$k_1 \delta \le \tau_k \le k_2 \delta. \tag{7}$$

Then incrementally update the model :

$$\begin{aligned}
\widetilde{f_n}(\xi, u) &= \frac{1}{n} \sum\nolimits_{k=1}^{n} \frac{y_k - x_k}{\tau_k} \\
\widetilde{a_n}(\xi, u) &= \frac{1}{n} \sum\nolimits_{k=1}^{n} \frac{\left(y_k - x_k - \tau_k.\widetilde{f_n}(\xi, u)\right)\left(y_k - x_k - \tau_k.\widetilde{f_n}(\xi, u)\right)'}{\tau_k} \\
\widetilde{r}(\xi, u) &= \frac{1}{n} \sum_{k=1}^{n} r(x_k, u)
\end{aligned} \tag{8}$$

and compute the approximated time $\widetilde{\tau}(x, u)$ and the approximated probabilities of transition $\widetilde{p}(\xi, u, \zeta)$ by replacing $f$ and $a$ by $\widetilde{f}$ and $\widetilde{a}$ in (5) and (6).

We obtain the following updating rule of the $V^\delta$-value of state $\xi$ :

$$V_{n+1}^\delta(\xi) = \sup_{u \in U^\delta} \left[ \gamma^{\widetilde{\tau}(x,u)} \sum\nolimits_\zeta \widetilde{p}(\xi, u, \zeta) V_n^\delta(\zeta) + \widetilde{\tau}(x, u)\widetilde{r}(\xi, u) \right] \tag{9}$$

which can be used as an off-line (synchronous, Gauss-Seidel, asynchronous) or on-time (for example by updating $V_n^\delta(\xi)$ as soon as a trajectory exits from the neighbourood of $\xi$) DP algorithm (see [BBS95]).

Besides, when a trajectory hits the boundary $\partial O$ at some exit point $x_k(\tau)$ then update the closest state $\xi \in \partial \Sigma^\delta$ with :

$$V_n^\delta(\xi) = R(x_k(\tau)) \tag{10}$$

**Theorem 3 (Convergence of the algorithm)** *Suppose that the model as well as the $V^\delta$-value of every state $\xi \in \Sigma^\delta$ and control $u \in U^\delta$ are regularly updated (respectively with (8) and (9)) and that every state $\xi \in \partial \Sigma^\delta$ are updated with (10) at least once. Then $\forall \varepsilon > 0, \exists \Delta$ such that $\forall \delta \le \Delta, \exists N, \forall n \ge N,$*

$$\sup_{\xi \in \Sigma^\delta} |V_n^\delta(\xi) - V(\xi)| \le \varepsilon \text{ with probability 1}$$

## 4   Conclusion

This paper presents a model-based RL algorithm for continuous stochastic control problems. A model of the dynamics is approximated by the mean and the covariance of successive states. Then, a RL updating rule based on a convergent FD scheme is deduced and in the hypothesis of an adequate exploration, the convergence to the optimal solution is proved as the discretization step $\delta$ tends to 0 and the number of iteration tends to infinity. This result is to be compared to the model-free RL algorithm for the deterministic case in [Mun97]. An interesting possible future work should be to consider model-free algorithms in the stochastic case for which a $Q$-learning rule (see [Wat89]) could be relevant.

## A   Appendix: proof of the convergence

Let $M_f, M_a, M_{f_x}$ and $M_{\sigma_x}$ be the upper bounds of $f, a, f_x$ and $\sigma_x$ and $m_f$ the lower bound of $f$. Let $E^\delta = \sup_{\xi \in \Sigma^\delta} |V^\delta(\xi) - V(\xi)|$ and $E_n^\delta = \sup_{\xi \in \Sigma^\delta} |V_n^\delta(\xi) - V^\delta(\xi)|$.

### A.1   Estimation error of the model $\widetilde{f_n}$ and $\widetilde{a_n}$ and the probabilities $\widetilde{p}_n$

Suppose that the trajectory $x_k(t)$ occured for some occurence $w_k(t)$ of the brownian motion: $x_k(t) = x_k + \int_{t_k}^t f(x_k(t), u)dt + \int_{t_k}^t \sigma(x_k(t), u)dw_k$. Then we consider a trajectory $z_k(t)$ starting from $\xi$ at $t_k$ and following the same brownian motion: $z_k(t) = \xi + \int_{t_k}^t f(z_k(t), u)dt + \int_{t_k}^t \sigma(z_k(t), u)dw_k$.

Let $z_k = z_k(t_k + \tau_k)$. Then $(y_k - x_k) - (z_k - \xi) = \int_{t_k} [f(x_k(t), u) - f(z_k(t), u)] dt + \int_{t_k}^{t_k + \tau_k} [\sigma(x_k(t), u) - \sigma(z_k(t), u)] dw_k$. Thus, from the $\mathcal{C}^1$ property of $f$ and $\sigma$,

$$\|(y_k - x_k) - (z_k - \xi)\| \le (M_{f_x} + M_{\sigma_x}).k_N.\tau_k.\delta. \tag{11}$$

The diffusion processes has the following property (see for example the Itô-Taylor majoration in [KP95]): $E_x[z_k] = \xi + \tau_k.f(\xi, u) + O(\tau_k^2)$ which, from (7), is equivalent to: $E_x\left[\frac{z_k - \xi}{\tau_k}\right] = f(\xi, u) + O(\delta)$. Thus from the law of large numbers and (11):

$$\limsup_{n \to \infty} \left\|\widetilde{f_n}(\xi, u) - f(\xi, u)\right\| = \limsup_{n \to \infty} \left\|\frac{1}{n} \sum_{k=1}^n \left[\frac{y_k - x_k}{\tau_k} - \frac{z_k - \xi}{\tau_k}\right]\right\| + O(\delta)$$

$$= (M_{f_x} + M_{\sigma_x}).k_N.\delta + O(\delta) = O(\delta) \text{ w.p. } 1 \tag{12}$$

Besides, diffusion processes have the following property (again see [KP95]): $E_x\left[(z_k - \xi)(z_k - \xi)'\right] = a(\xi, u)\tau_k + f(\xi, u).f(\xi, u)'.\tau_k^2 + O(\tau_k^3)$ which, from (7), is equivalent to: $E_x\left[\frac{(z_k - \xi - \tau_k f(\xi, u))(z_k - \xi - \tau_k f(\xi, u))'}{\tau_k}\right] = a(\xi, u) + O(\delta^2)$. Let $r_k = z_k - \xi - \tau_k f(\xi, u)$ and $\widetilde{r}_k = y_k - x_k - \tau_k \widetilde{f_n}(\xi, u)$ which satisfy (from (11) and (12)):

$$\|r_k - \widetilde{r}_k\| = (M_{f_x} + M_{\sigma_x}).\tau_k.k_N.\delta + \tau_k.O(\delta) \tag{13}$$

From the definition of $\widetilde{a_n}(\xi, u)$, we have: $\widetilde{a_n}(\xi, u) - a(\xi, u) = \frac{1}{n} \sum_{k=1}^n \frac{\widetilde{r}_k.\widetilde{r}_k'}{\tau_k} - E_x\left[\frac{r_k.r_k'}{\tau_k}\right] + O(\delta^2)$ and from the law of large numbers, (12) and (13), we have:

$$\limsup_{n \to \infty} \|\widetilde{a_n}(\xi, u) - a(\xi, u)\| = \limsup_{n \to \infty} \left\|\frac{1}{n} \sum_{k=1}^n \frac{\widetilde{r}_k.\widetilde{r}_k'}{\tau_k} - \frac{r_k.r_k'}{\tau_k}\right\| + O(\delta^2)$$

$$= \|\widetilde{r}_k - r_k\| \limsup_{n \to \infty} \frac{1}{n} \sum_{k=1}^n \left(\left\|\frac{\widetilde{r}_k}{\tau_k}\right\| + \left\|\frac{r_k}{\tau_k}\right\|\right) + O(\delta^2) = O(\delta^2)$$

with probability 1. Thus there exists $k_f$ and $k_a$ s.t. $\exists \Delta_1, \forall \delta \leq \Delta_1, \exists N_1, n \geq N_1,$

$$\begin{aligned} \left\| \widetilde{f_n}(\xi, u) - f(\xi, u) \right\| &\leq k_f.\delta \text{ w.p. } 1 \\ \left\| \widetilde{a_n}(\xi, u) - a(\xi, u) \right\| &\leq k_a.\delta^2 \text{ w.p. } 1 \end{aligned} \tag{14}$$

Besides, from (5) and (14), we have:

$$|\tau(\xi, u) - \widetilde{\tau}_n(\xi, u)| \leq \frac{d.(k_f.\delta^2 + d.k_a\delta^2)}{(d.m_f.\delta)^2} \delta^2 \leq k_\tau.\delta^2 \tag{15}$$

and from a property of exponential function,

$$\left| \gamma^{\tau(\xi, u)} - \gamma^{\widetilde{\tau}_n(\xi, u)} \right| = k_\tau. \ln \frac{1}{\gamma}.\delta^2. \tag{16}$$

We can deduce from (14) that:

$$\limsup_{n \to \infty} |p(\xi, u, \zeta) - \widetilde{p_n}(\xi, u, \zeta)| \leq \frac{(2.\delta.M_f + d.M_a)(2.k_f + d.k_a)\delta^2}{\delta m_f - (2.k_f + d.k_a)\delta^2} \leq k_p\delta \text{ w.p. } 1 \tag{17}$$

with $k_p = 4(d.M_a)(2.k_f + d.k_a)$ for $\delta \leq \Delta_2 = \min\left\{ \frac{m_f}{2.k_f + d.k_a}, \frac{d.M_a}{2.\delta.M_f} \right\}.$

## A.2   Estimation of $|V_{n+1}^\delta(\xi) - V^\delta(\xi)|$

After having updated $V_n^\delta(\xi)$ with rule (9), let $\Lambda$ denote the difference $|V_{n+1}^\delta(\xi) - V^\delta(\xi)|$. From (4), (9) and (8),

$$\begin{aligned} \Lambda \leq\ & \gamma^{\tau(\xi, u)} \sum_\zeta [p(\xi, u, \zeta) - \widetilde{p}(\xi, u, \zeta)] V^\delta(\zeta) + \left( \gamma^{\tau(\xi, u)} - \gamma^{\widetilde{\tau}(\xi, u)} \right) \sum_\zeta \widetilde{p}(\xi, u, \zeta) V^\delta(\zeta) \\ & + \gamma^{\widetilde{\tau}(\xi, u)}. \sum_\zeta \widetilde{p}(\xi, u, \zeta) \left[ V^\delta(\zeta) - V_n^\delta(\zeta) \right] + \sum_\zeta \widetilde{p}(\xi, u, \zeta).\widetilde{\tau}(\xi, u) \left[ r(\xi, u) - \widetilde{r}(\xi, u) \right] \\ & + \sum_\zeta \widetilde{p}(\xi, u, \zeta) \left[ \widetilde{\tau}(\xi, u) - \tau(\xi, u) \right] r(\xi, u) \text{ for all } u \in U^\delta \end{aligned}$$

As $V$ is differentiable we have : $V(\zeta) = V(\xi) + V_x.(\zeta - \xi) + o(\|\zeta - \xi\|)$. Let us define a linear function $\widetilde{V}$ such that: $\widetilde{V}(x) = V(\xi) + V_x.(x - \xi)$. Then we have: $[p(\xi, u, \zeta) - \widetilde{p}(\xi, u, \zeta)] V^\delta(\zeta) = [p(\xi, u, \zeta) - \widetilde{p}(\xi, u, \zeta)].[V^\delta(\zeta) - V(\zeta)] + [p(\xi, u, \zeta) - \widetilde{p}(\xi, u, \zeta)] V(\zeta)$, thus: $\sum_\zeta [p(\xi, u, \zeta) - \widetilde{p}(\xi, u, \zeta)] V^\delta(\zeta) = k_p.E^\delta.\delta + \sum_\zeta [p(\xi, u, \zeta) - \widetilde{p}(\xi, u, \zeta)] [\widetilde{V}(\zeta) + o(\delta)] = [\widetilde{V}(\eta) - \widetilde{V}(\widetilde{\eta})] + k_p.E^\delta.\delta + o(\delta) = [\widetilde{V}(\eta) - \widetilde{V}(\widetilde{\eta})] + o(\delta)$ with: $\eta = \sum_\zeta p(\xi, u, \zeta)(\zeta - \xi)$ and $\widetilde{\eta} = \sum_\zeta \widetilde{p}(\xi, u, \zeta)(\zeta - \xi)$. Besides, from the convergence of the scheme (theorem 2), we have $E^\delta.\delta = o(\delta)$. From the linearity of $\widetilde{V}$, $\left| \widetilde{V}(\zeta) - \widetilde{V}(\widetilde{\zeta}) \right| \leq \left\| \zeta - \widetilde{\zeta} \right\|.M_{V_x} \leq 2k_p\delta^2$. Thus $\left| \sum_\zeta [p(\xi, u, \zeta) - \widetilde{p}(\xi, u, \zeta)] V^\delta(\zeta) \right| = o(\delta)$ and from (15), (16) and the Lipschitz property of $r$,

$$\Lambda = \left| \gamma^{\widetilde{\tau}(\xi, u)}. \sum_\zeta \widetilde{p}(\xi, u, \zeta) \left[ V^\delta(\zeta) - V_n^\delta(\zeta) \right] \right| + o(\delta).$$

As $\gamma^{\widetilde{\tau}(\xi, u)} \leq 1 - \frac{\widetilde{\tau}(\xi, u)}{2} \ln \frac{1}{\gamma} \leq 1 - \frac{\tau(\xi, u) - k_\tau \delta^2}{2} \ln \frac{1}{\gamma} \leq 1 - \left( \frac{\delta}{2d(M_f + d.M_a)} - \frac{k_\tau}{2}\delta^2 \right) \ln \frac{1}{\gamma},$ we have:

$$\Lambda = (1 - k.\delta)E_n^\delta + o(\delta) \tag{18}$$

with $k = \frac{1}{2d(M_f + d.M_a)}.$

## A.3 A sufficient condition for $\sup_{\xi \in \Sigma^\delta} \left| V_n^\delta(\xi) - V^\delta(\xi) \right| \leq \varepsilon_2$

Let us suppose that for all $\xi \in \Sigma^\delta$, the following conditions hold for some $\alpha > 0$

$$E_n^\delta > \varepsilon_2 \Rightarrow \left| V_{n+1}^\delta(\xi) - V^\delta(\xi) \right| \leq E_n^\delta - \alpha \qquad (19)$$

$$E_n^\delta \leq \varepsilon_2 \Rightarrow \left| V_{n+1}^\delta(\xi) - V^\delta(\xi) \right| \leq \varepsilon_2 \qquad (20)$$

From the hypothesis that all states $\xi \in \Sigma^\delta$ are regularly updated, there exists an integer $m$ such that at stage $n + m$ all the $\xi \in \Sigma^\delta$ have been updated at least once since stage $n$. Besides, since all $\xi \in \partial G^\delta$ are updated at least once with rule (10), $\forall \xi \in \partial G^\delta, |V_n^\delta(\xi) - V^\delta(\xi)| = |R(x_\xi(\tau)) - R(\xi)| \leq 2.L_R.\delta \leq \varepsilon_2$ for any $\delta \leq \Delta_3 = \frac{\varepsilon_2}{2.L_R}$. Thus, from (19) and (20) we have:

$$E_n^\delta > \varepsilon_2 \Rightarrow E_{n+m}^\delta \leq E_n^\delta - \alpha$$

$$E_n^\delta \leq \varepsilon_2 \Rightarrow E_{n+m}^\delta \leq \varepsilon_2$$

Thus there exists $N$ such that : $\forall n \geq N, E_n^\delta \leq \varepsilon_2$.

## A.4 Convergence of the algorithm

Let us prove theorem 3. For any $\varepsilon > 0$, let us consider $\varepsilon_1 > 0$ and $\varepsilon_2 > 0$ such that $\varepsilon_1 + \varepsilon_2 = \varepsilon$. Assume $E_n^\delta > \varepsilon_2$, then from (18), $\Lambda = E_n^\delta - k.\delta.\varepsilon_2 + o(\delta) \leq E_n^\delta - k.\delta.\frac{\varepsilon_2}{2}$ for $\delta \leq \Delta_3$. Thus (19) holds for $\alpha = k.\delta.\frac{\varepsilon_2}{2}$. Suppose now that $E_n^\delta \leq \varepsilon_2$. From (18), $\Lambda \leq (1 - k.\delta)\varepsilon_2 + o(\delta) \leq \varepsilon_2$ for $\delta \leq \Delta_3$ and condition (20) is true.

Thus for $\delta \leq \min\{\Delta_1, \Delta_2, \Delta_3\}$, the sufficient conditions (19) and (20) are satisfied. So there exists $N$, for all $n \geq N$, $E_n^\delta \leq \varepsilon_2$. Besides, from the convergence of the scheme (theorem 2), there exists $\Delta_0$ st. $\forall \delta \leq \Delta_0, \sup_{\xi \in \Sigma^\delta} |V^\delta(\xi) - V(\xi)| \leq \varepsilon_1$.

Thus for $\delta \leq \min\{\Delta_0, \Delta_1, \Delta_2, \Delta_3\}, \exists N, \forall n \geq N,$

$$\sup_{\xi \in \Sigma^\delta} |V_n^\delta(\xi) - V(\xi)| \leq \sup_{\xi \in \Sigma^\delta} |V_n^\delta(\xi) - V^\delta(\xi)| + \sup_{\xi \in \Sigma^\delta} |V^\delta(\xi) - V(\xi)| \leq \varepsilon_1 + \varepsilon_2 = \varepsilon.$$

# References

[BBS95] Andrew G. Barto, Steven J. Bradtke, and Satinder P. Singh. Learning to act using real-time dynamic programming. *Artificial Intelligence*, (72):81–138, 1995.

[Ber87] Dimitri P. Bertsekas. *Dynamic Programming : Deterministic and Stochastic Models*. Prentice Hall, 1987.

[FS93] Wendell H. Fleming and H. Mete Soner. *Controlled Markov Processes and Viscosity Solutions*. Applications of Mathematics. Springer-Verlag, 1993.

[KP95] Peter E. Kloeden and Eckhard Platen. *Numerical Solutions of Stochastic Differential Equations*. Springer-Verlag, 1995.

[Kry80] N.V. Krylov. *Controlled Diffusion Processes*. Springer-Verlag, New York, 1980.

[Kus90] Harold J. Kushner. Numerical methods for stochastic control problems in continuous time. *SIAM J. Control and Optimization*, 28:999–1048, 1990.

[Mun97] Rémi Munos. A convergent reinforcement learning algorithm in the continuous case based on a finite difference method. *International Joint Conference on Artificial Intelligence*, 1997.

[Wat89] Christopher J.C.H. Watkins. *Learning from delayed reward*. PhD thesis, Cambridge University, 1989.
